# Dynamic Pooling and Unfolding Recursive Autoencoders for Paraphrase Detection

**Richard Socher, Eric H. Huang, Jeffrey Pennington**∗**, Andrew Y. Ng, Christopher D. Manning**
Computer Science Department, Stanford University, Stanford, CA 94305, USA
∗SLAC National Accelerator Laboratory, Stanford University, Stanford, CA 94309, USA
richard@socher.org, {ehhuang,jpennin,ang,manning}@stanford.edu

## Abstract

Paraphrase detection is the task of examining two sentences and determining whether they have the same meaning. In order to obtain high accuracy on this task, thorough syntactic and semantic analysis of the two statements is needed. We introduce a method for paraphrase detection based on recursive autoencoders (RAE). Our unsupervised RAEs are based on a novel unfolding objective and learn feature vectors for phrases in syntactic trees. These features are used to measure the word- and phrase-wise similarity between two sentences. Since sentences may be of arbitrary length, the resulting matrix of similarity measures is of variable size. We introduce a novel dynamic pooling layer which computes a fixed-sized representation from the variable-sized matrices. The pooled representation is then used as input to a classifier. Our method outperforms other state-of-the-art approaches on the challenging MSRP paraphrase corpus.

## 1   Introduction

Paraphrase detection determines whether two phrases of arbitrary length and form capture the same meaning. Identifying paraphrases is an important task that is used in information retrieval, question answering [1], text summarization, plagiarism detection [2] and evaluation of machine translation [3], among others. For instance, in order to avoid adding redundant information to a summary one would like to detect that the following two sentences are paraphrases:

  S1  The judge also refused to postpone the trial date of Sept. 29.
  S2  Obus also denied a defense motion to postpone the September trial date.

We present a joint model that incorporates the similarities between both single word features as well as multi-word phrases extracted from the nodes of parse trees. Our model is based on two novel components as outlined in Fig. 1. The first component is an *unfolding recursive autoencoder* (RAE) for unsupervised feature learning from unlabeled parse trees. The RAE is a recursive neural network. It learns feature representations for each node in the tree such that the word vectors underneath each node can be recursively reconstructed.

These feature representations are used to compute a similarity matrix that compares both the single words as well as all nonterminal node features in both sentences. In order to keep as much of the resulting global information of this comparison as possible and deal with the arbitrary length of the two sentences, we then introduce our second component: a new *dynamic pooling layer which outputs a fixed-size representation*. Any classifier such as a softmax classifier can then be used to classify whether the two sentences are paraphrases or not.

We first describe the unsupervised feature learning with RAEs followed by a description of pooling and classification. In experiments we show qualitative comparisons of different RAE models and describe our state-of-the-art results on the Microsoft Research Paraphrase (MSRP) Corpus introduced by Dolan et al. [4]. Lastly, we discuss related work.

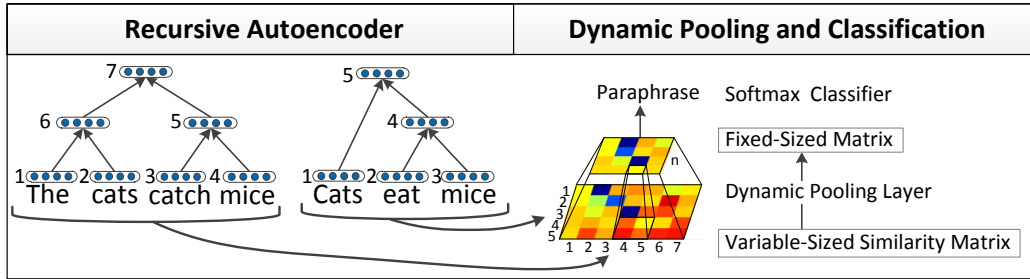

Figure 1: An overview of our paraphrase model. The recursive autoencoder learns phrase features for each node in a parse tree. The distances between all nodes then fill a similarity matrix whose size depends on the length of the sentences. Using a novel dynamic pooling layer we can compare the variable-sized sentences and classify pairs as being paraphrases or not.

## 2 Recursive Autoencoders

In this section we describe two variants of unsupervised recursive autoencoders which can be used to learn features from parse trees. The RAE aims to find vector representations for variable-sized phrases spanned by each node of a parse tree. These representations can then be used for subsequent supervised tasks. Before describing the RAE, we briefly review neural language models which compute word representations that we give as input to our algorithm.

### 2.1 Neural Language Models

The idea of neural language models as introduced by Bengio et al. [5] is to jointly learn an embedding of words into an $n$-dimensional vector space and to use these vectors to predict how likely a word is given its context. Collobert and Weston [6] introduced a new neural network model to compute such an embedding. When these networks are optimized via gradient ascent the derivatives modify the word embedding matrix $L \in \mathbb{R}^{n \times |V|}$, where $|V|$ is the size of the vocabulary. The word vectors inside the embedding matrix capture distributional syntactic and semantic information via the word's co-occurrence statistics. For further details and evaluations of these embeddings, see [5, 6, 7, 8].

Once this matrix is learned on an unlabeled corpus, we can use it for subsequent tasks by using each word's vector (a column in $L$) to represent that word. In the remainder of this paper, we represent a sentence (or any $n$-gram) as an ordered list of these vectors $(x_1, \ldots, x_m)$. This word representation is better suited for autoencoders than the binary number representations used in previous related autoencoder models such as the recursive autoassociative memory (RAAM) model of Pollack [9, 10] or recurrent neural networks [11] since the activations are inherently continuous.

### 2.2 Recursive Autoencoder

Fig. 2 (left) shows an instance of a recursive autoencoder (RAE) applied to a given parse tree as introduced by [12]. Unlike in that work, here we assume that such a tree is given for each sentence by a parser. Initial experiments showed that having a syntactically plausible tree structure is important for paraphrase detection. Assume we are given a list of word vectors $x = (x_1, \ldots, x_m)$ as described in the previous section. The binary parse tree for this input is in the form of branching triplets of parents with children: $(p \rightarrow c_1 c_2)$. The trees are given by a syntactic parser. Each child can be either an input word vector $x_i$ or a nonterminal node in the tree. For both examples in Fig. 2, we have the following triplets: $((y_1 \rightarrow x_2 x_3), (y_2 \rightarrow x_1 y_1)), \forall x, y \in \mathbb{R}^n$.

Given this tree structure, we can now compute the parent representations. The first parent vector $p = y_1$ is computed from the children $(c_1, c_2) = (x_2, x_3)$ by one standard neural network layer:

$$p \quad = \quad f(W_e[c_1; c_2] + b), \tag{1}$$

where $[c_1; c_2]$ is simply the concatenation of the two children, $f$ an element-wise activation function such as $\tanh$ and $W_e \in \mathbb{R}^{n \times 2n}$ the encoding matrix that we want to learn. One way of assessing how well this $n$-dimensional vector represents its direct children is to decode their vectors in a

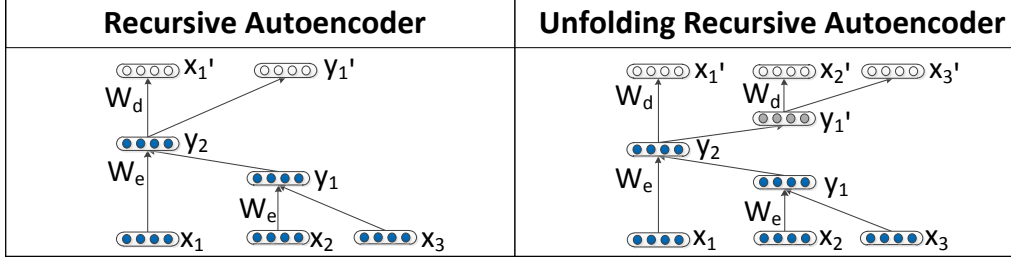

Figure 2: Two autoencoder models with details of the reconstruction at node $y_2$. For simplicity we left out the reconstruction layer at the first node $y_1$ which is the same standard autoencoder for both models. Left: A standard autoencoder that tries to reconstruct only its direct children. Right: The unfolding autoencoder which tries to reconstruct all leaf nodes underneath each node.

reconstruction layer and then to compute the Euclidean distance between the original input and its reconstruction:

$$[c_1'; c_2'] \;\; = \;\; f(W_d p + b_d) \qquad E_{rec}(p) = ||[c_1; c_2] - [c_1'; c_2']||^2 \,. \tag{2}$$

In order to apply the autoencoder recursively, the same steps repeat. Now that $y_1$ is given, we can use Eq. 1 to compute $y_2$ by setting the children to be $(c_1, c_2) = (x_1, y_1)$. Again, after computing the intermediate parent vector $p = y_2$, we can assess how well this vector captures the content of the children by computing the reconstruction error as in Eq. 2. The process repeats until the full tree is constructed and each node has an associated reconstruction error.

During training, the goal is to minimize the reconstruction error of all input pairs at nonterminal nodes $p$ in a given parse tree $\mathcal{T}$:

$$E_{rec}(\mathcal{T}) = \sum_{p \in \mathcal{T}} E_{rec}(p) \tag{3}$$

For the example in Fig. 2 (left), we minimize $E_{rec}(\mathcal{T}) = E_{rec}(y_1) + E_{rec}(y_2)$.

Since the RAE computes the hidden representations it then tries to reconstruct, it could potentially lower reconstruction error by shrinking the norms of the hidden layers. In order to prevent this, we add a length normalization layer $p = p/||p||$ to this RAE model (referred to as the standard RAE). Another more principled solution is to use a model in which each node tries to reconstruct its entire subtree and then measure the reconstruction of the original leaf nodes. Such a model is described in the next section.

## 2.3 Unfolding Recursive Autoencoder

The unfolding RAE has the same encoding scheme as the standard RAE. The difference is in the decoding step which tries to reconstruct the entire spanned subtree underneath each node as shown in Fig. 2 (right). For instance, at node $y_2$, the reconstruction error is the difference between the leaf nodes underneath that node $[x_1; x_2; x_3]$ and their reconstructed counterparts $[x_1'; x_2'; x_3']$. The unfolding produces the reconstructed leaves by starting at $y_2$ and computing

$$[x_1'; y_1'] = f(W_d y_2 + b_d). \tag{4}$$

Then it recursively splits $y_1'$ again to produce vectors

$$[x_2'; x_3'] = f(W_d y_1' + b_d). \tag{5}$$

In general, we repeatedly use the decoding matrix $W_d$ to unfold each node with the same tree structure as during encoding. The reconstruction error is then computed from a concatenation of the word vectors in that node's span. For a node $y$ that spans words $i$ to $j$:

$$E_{rec}(y_{(i,j)}) = \left|\left|[x_i; \dots; x_j] - [x_i'; \dots; x_j']\right|\right|^2 \,. \tag{6}$$

The unfolding autoencoder essentially tries to encode each hidden layer such that it best reconstructs its entire subtree to the leaf nodes. Hence, it will not have the problem of hidden layers shrinking in norm. Another potential problem of the standard RAE is that it gives equal weight to the last merged phrases even if one is only a single word (in Fig. 2, $x_1$ and $y_1$ have similar weight in the last merge). In contrast, the unfolding RAE captures the increased importance of a child when the child represents a larger subtree.

## 2.4 Deep Recursive Autoencoder

Both types of RAE can be extended to have multiple encoding layers at each node in the tree. Instead of transforming both children directly into parent $p$, we can have another hidden layer $h$ in between. While the top layer at each node has to have the same dimensionality as each child (in order for the same network to be recursively compatible), the hidden layer may have arbitrary dimensionality. For the two-layer encoding network, we would replace Eq. 1 with the following:

$$h = f(W_e^{(1)}[c_1; c_2] + b_e^{(1)}) \tag{7}$$

## 2.5 RAE Training

$$p = f(W_e^{(2)}h + b_e^{(2)}). \tag{8}$$

For training we use a set of parse trees and then minimize the sum of all nodes' reconstruction errors. We compute the gradient efficiently via backpropagation through structure [13]. Even though the objective is not convex, we found that L-BFGS run with mini-batch training works well in practice. Convergence is smooth and the algorithm typically finds a good locally optimal solution.

After the unsupervised training of the RAE, we demonstrate that the learned feature representations capture syntactic and semantic similarities and can be used for paraphrase detection.

# 3 An Architecture for Variable-Sized Similarity Matrices

Now that we have described the unsupervised feature learning, we explain how to use these features to classify sentence pairs as being in a paraphrase relationship or not.

## 3.1 Computing Sentence Similarity Matrices

Our method incorporates both single word and phrase similarities in one framework. First, the RAE computes phrase vectors for the nodes in a given parse tree. We then compute Euclidean distances between all word and phrase vectors of the two sentences. These distances fill a similarity matrix $\mathcal{S}$ as shown in Fig. 1. For computing the similarity matrix, the rows and columns are first filled by the words in their original sentence order. We then add to each row and column the nonterminal nodes in a depth-first, right-to-left order.

Simply extracting aggregate statistics of this table such as the average distance or a histogram of distances cannot accurately capture the global structure of the similarity comparison. For instance, paraphrases often have low or zero Euclidean distances in elements close to the diagonal of the similarity matrix. This happens when similar words align well between the two sentences. However, since the matrix dimensions vary based on the sentence lengths one cannot simply feed the similarity matrix into a standard neural network or classifier.

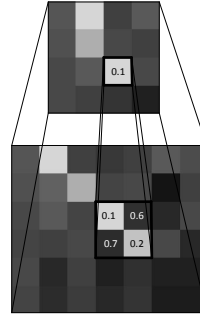

Figure 3: Example of the dynamic min-pooling layer finding the smallest number in a pooling window region of the original similarity matrix $\mathcal{S}$.

## 3.2 Dynamic Pooling

Consider a similarity matrix $\mathcal{S}$ generated by sentences of lengths $n$ and $m$. Since the parse trees are binary and we also compare all nonterminal nodes, $\mathcal{S} \in \mathbb{R}^{(2n-1) \times (2m-1)}$. We would like to map $S$ into a matrix $S_{\text{pooled}}$ of fixed size, $n_p \times n_p$. Our first step in constructing such a map is to partition the rows and columns of $\mathcal{S}$ into $n_p$ roughly equal parts, producing an $n_p \times n_p$ grid.[1] We then define $S_{\text{pooled}}$ to be the matrix of minimum values of each rectangular region within this grid, as shown in Fig. 3.

The matrix $S_{\text{pooled}}$ loses some of the information contained in the original similarity matrix but it still captures much of its global structure. Since elements of $S$ with small Euclidean distances show that

| Center Phrase | Recursive Average | RAE | Unfolding RAE |
|---|---|---|---|
| the U.S. | the U.S. and German | the Swiss | the former U.S. |
| suffering low morale | suffering a 1.9 billion baht UNK 76 million | suffering due to no fault of my own | suffering heavy casualties |
| to watch hockey | to watch one Jordanian border policeman stamp the Israeli passports | to watch television | to watch a video |
| advance to the next round | advance to final qualifying round in Argentina | advance to the final of the UNK 1.1 million Kremlin Cup | advance to the semis |
| a prominent political figure | such a high-profile figure | the second high-profile opposition figure | a powerful business figure |
| Seventeen people were killed | "Seventeen people were killed, including a prominent politician " | Fourteen people were killed | Fourteen people were killed |
| conditions of his release | "conditions of peace, social stability and political harmony " | conditions of peace, social stability and political harmony | negotiations for their release |

Table 1: Nearest neighbors of randomly chosen phrases. Recursive averaging and the standard RAE focus mostly on the last merged words and incorrectly add extra information. The unfolding RAE captures most closely both syntactic and semantic similarities.

there are similar words or phrases in both sentences, we keep this information by applying a $\min$ function to the pooling regions. Other functions, like averaging, are also possible, but might obscure the presence of similar phrases. This dynamic pooling layer could make use of overlapping pooling regions, but for simplicity, we consider only non-overlapping pooling regions. After pooling, we normalize each entry to have 0 mean and variance 1.

## 4 Experiments

For unsupervised RAE training we used a subset of 150,000 sentences from the NYT and AP sections of the Gigaword corpus. We used the Stanford parser [14] to create the parse trees for all sentences. For initial word embeddings we used the 100-dimensional vectors computed via the unsupervised method of Collobert and Weston [6] and provided by Turian et al. [8].

For all paraphrase experiments we used the Microsoft Research paraphrase corpus (MSRP) introduced by Dolan et al. [4]. The dataset consists of 5,801 sentence pairs. The average sentence length is 21, the shortest sentence has 7 words and the longest 36. 3,900 are labeled as being in the paraphrase relationship (technically defined as "mostly bidirectional entailment"). We use the standard split of 4,076 training pairs (67.5% of which are paraphrases) and 1,725 test pairs (66.5% paraphrases). All sentences were labeled by two annotators who agreed in 83% of the cases. A third annotator resolved conflicts. During dataset collection, negative examples were selected to have high lexical overlap to prevent trivial examples. For more information see [4, 15].

As described in Sec. 2.4, we can have deep RAE networks with two encoding or decoding layers. The hidden RAE layer (see $h$ in Eq. 8) was set to have 200 units for both standard and unfolding RAEs.

### 4.1 Qualitative Evaluation of Nearest Neighbors

In order to show that the learned feature representations capture important semantic and syntactic information even for higher nodes in the tree, we visualize nearest neighbor phrases of varying length. After embedding sentences from the Gigaword corpus, we compute nearest neighbors for all nodes in all trees. In Table 1 the first phrase is a randomly chosen phrase and the remaining phrases are the closest phrases in the dataset that are not in the same sentence. We use Euclidean distance between the vector representations. Note that we do not constrain the neighbors to have the same word length. We compare the two autoencoder models above: RAE and unfolding RAE without hidden layers, as well as a recursive averaging baseline (R.Avg). R.Avg recursively takes the average of both child vectors in the syntactic tree. We only report results of RAEs without hidden layers between the children and parent vectors. Even though the deep RAE networks have more parameters to learn complex encodings they do not perform as well in this and the next task. This is likely due to the fact that they get stuck in local optima during training.

| Encoding Input | Generated Text from Unfolded Reconstruction |
|---|---|
| a December summit | a December summit |
| the first qualifying session | the first qualifying session |
| English premier division club | Irish presidency division club |
| the safety of a flight | the safety of a flight |
| the signing of the accord | the signing of the accord |
| the U.S. House of Representatives | the U.S. House of Representatives |
| enforcement of the economic embargo | enforcement of the national embargo |
| visit and discuss investment possibilities | visit and postpone financial possibilities |
| the agreement it made with Malaysia | the agreement it made with Malaysia |
| the full bloom of their young lives | the lower bloom of their democratic lives |
| the organization for which the men work | the organization for Romania the reform work |
| a pocket knife was found in his suitcase in the plane's cargo hold | a bomb corpse was found in the mission in the Irish car language case |

Table 2: Original inputs and generated output from unfolding and reconstruction. Words are the nearest neighbors to the reconstructed leaf node vectors. The unfolding RAE can reconstruct perfectly almost all phrases of 2 and 3 words and many with up to 5 words. Longer phrases start to get incorrect nearest neighbor words. For the standard RAE good reconstructions are only possible for two words. Recursive averaging cannot recover any words.

Table 1 shows several interesting phenomena. Recursive averaging is almost entirely focused on an exact string match of the last merged words of the current phrase in the tree. This leads the nearest neighbors to incorrectly add various extra information which would break the paraphrase relationship if we only considered the top node vectors and ignores syntactic similarity. The standard RAE does well though it is also somewhat focused on the last merges in the tree. Finally, the unfolding RAE captures most closely the underlying syntactic and semantic structure.

## 4.2 Reconstructing Phrases via Recursive Decoding

In this section we analyze the information captured by the unfolding RAE's 100-dimensional phrase vectors. We show that these 100-dimensional vector representations can not only capture and memorize single words but also longer, unseen phrases.

In order to show how much of the information can be recovered we recursively reconstruct sentences after encoding them. The process is similar to unfolding during training. It starts from a phrase vector of a nonterminal node in the parse tree. We then unfold the tree as given during encoding and find the nearest neighbor word to each of the reconstructed leaf node vectors. Table 2 shows that the unfolding RAE can very well reconstruct phrases of up to length five. No other method that we compared had such reconstruction capabilities. Longer phrases retain some correct words and usually the correct part of speech but the semantics of the words get merged. The results are from the unfolding RAE that directly computes the parent representation as in Eq. 1.

## 4.3 Evaluation on Full-Sentence Paraphrasing

We now turn to evaluating the unsupervised features and our dynamic pooling architecture in our main task of paraphrase detection.

Methods which are based purely on vector representations invariably lose some information. For instance, numbers often have very similar representations, but even small differences are crucial to reject the paraphrase relation in the MSRP dataset. Hence, we add three number features. The first is $1$ if two sentences contain exactly the same numbers or no number and $0$ otherwise, the second is $1$ if both sentences contain the same numbers and the third is $1$ if the set of numbers in one sentence is a strict subset of the numbers in the other sentence. Since our pooling-layer cannot capture sentence length or the number of exact string matches, we also add the difference in sentence length and the percentage of words and phrases in one sentence that are in the other sentence and vice-versa. We also report performance without these three features (only $\mathcal{S}$).

For all of our models and training setups, we perform 10-fold cross-validation on the training set to choose the best regularization parameters and $n_p$, the size of the pooling matrix $\mathcal{S} \in \mathbb{R}^{n_p \times n_p}$. In our best model, the regularization for the RAE was $10^{-5}$ and $0.05$ for the softmax classifier. The best pooling size was consistently $n_p = 15$, slightly less than the average sentence length. For all sentence pairs $(S_1, S_2)$ in the training data, we also added $(S_2, S_1)$ to the training set in order to make the most use of the training data. This improved performance by $0.2\%$.

| Model | Acc. | F1 |
|---|---|---|
| All Paraphrase Baseline | 66.5 | 79.9 |
| Rus et al. (2008) [16] | 70.6 | 80.5 |
| Mihalcea et al. (2006) [17] | 70.3 | 81.3 |
| Islam and Inkpen (2007) [18] | 72.6 | 81.3 |
| Qiu et al. (2006) [19] | 72.0 | 81.6 |
| Fernando and Stevenson (2008) [20] | 74.1 | 82.4 |
| Wan et al. (2006) [21] | 75.6 | 83.0 |
| Das and Smith (2009) [15] | 73.9 | 82.3 |
| Das and Smith (2009) + 18 Features | 76.1 | 82.7 |
| Unfolding RAE + Dynamic Pooling | **76.8** | **83.6** |

Table 3: Test results on the MSRP paraphrase corpus. Comparisons of unsupervised feature learning methods (left), similarity feature extraction and supervised classification methods (center) and other approaches (right).

In our first set of experiments we compare several unsupervised feature learning methods: Recursive averaging as defined in Sec. 4.1, standard RAEs and unfolding RAEs. For each of the three methods, we cross-validate on the training data over all possible hyperparameters and report the best performance. We observe that the dynamic pooling layer is very powerful because it captures the global structure of the similarity matrix which in turn captures the syntactic and semantic similarities of the two sentences. With the help of this powerful dynamic pooling layer and good initial word vectors even the standard RAE and recursive averaging perform well on this dataset with an accuracy of 75.5% and 75.9% respectively. We obtain the best accuracy of 76.8% with the unfolding RAE without hidden layers. We tried adding 1 and 2 hidden encoding and decoding layers but performance only decreased by 0.2% and training became slower.

Next, we compare the dynamic pooling to simpler feature extraction methods. Our comparison shows that the dynamic pooling architecture is important for achieving high accuracy. For every setting we again exhaustively cross-validate on the training data and report the best performance. The settings and their accuracies are:

(i) $\mathcal{S}$-Hist: 73.0%. A histogram of values in the matrix $\mathcal{S}$. The low performance shows that our dynamic pooling layer better captures the global similarity information than aggregate statistics.

(ii) Only Feat: 73.2%. Only the three features described above. This shows that simple binary string and number matching can detect many of the simple paraphrases but fails to detect complex cases.

(iii) Only $\mathcal{S}_{pooled}$: 72.6%. Without the three features mentioned above. This shows that some information still gets lost in $\mathcal{S}_{pooled}$ and that a better treatment of numbers is needed. In order to better recover exact string matches it may be necessary to explore overlapping pooling regions.

(iv) Top Unfolding RAE Node: 74.2%. Instead of $\mathcal{S}_{pooled}$, use Euclidean distance between the two top sentence vectors. The performance shows that while the unfolding RAE is by itself very powerful, the dynamic pooling layer is needed to extract all information from its trees.

Table 3 shows our results compared to previous approaches (see next section). Our unfolding RAE and dynamic similarity pooling architecture achieves state-of-the-art performance without hand-designed semantic taxonomies and features such as WordNet. Note that the effective range of the accuracy lies between 66% (most frequent class baseline) and 83% (interannotator agreement).

In Table 4 we show several examples of correctly classified paraphrase candidate pairs together with their similarity matrix after dynamic min-pooling. The first and last pair are simple cases of paraphrase and not paraphrase. The second example shows a pooled similarity matrix when large chunks are swapped in both sentences. Our model is very robust to such transformations and gives a high probability to this pair. Even more complex examples such as the third with very few direct string matches (few blue squares) are correctly classified. The second to last example is highly interesting. Even though there is a clear diagonal with good string matches, the gap in the center shows that the first sentence contains much extra information. This is also captured by our model.

## 5 Related Work

The field of paraphrase detection has progressed immensely in recent years. Early approaches were based purely on lexical matching techniques [22, 23, 19, 24]. Since these methods are often based on exact string matches of $n$-grams, they fail to detect similar meaning that is conveyed by synonymous words. Several approaches [17, 18] overcome this problem by using Wordnet- and corpus-based semantic similarity measures. In their approach they choose for each open-class word the single most similar word in the other sentence. Fernando and Stevenson [20] improved upon this idea by computing a similarity matrix that captures all pair-wise similarities of single words in the two sentences. They then threshold the elements of the resulting similarity matrix and compute the mean

| L | Pr | Sentences | Sim.Mat. |
|---|---|---|---|
| P | 0.95 | (1) LLEYTON Hewitt yesterday traded his tennis racquet for his first sporting passion - Australian football - as the world champion relaxed before his Wimbledon title defence<br>(2) LLEYTON Hewitt yesterday traded his tennis racquet for his first sporting passion-Australian rules football-as the world champion relaxed ahead of his Wimbledon defence | 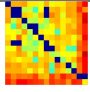 |
| P | 0.82 | (1) The lies and deceptions from Saddam have been well documented over 12 years<br>(2) It has been well documented over 12 years of lies and deception from Saddam | 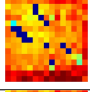 |
| P | 0.67 | (1) Pollack said the plaintiffs failed to show that Merrill and Blodget directly caused their losses<br>(2) Basically, the plaintiffs did not show that omissions in Merrill's research caused the claimed losses | 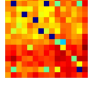 |
| N | 0.49 | (1) Prof Sally Baldwin, 63, from York, fell into a cavity which opened up when the structure collapsed at Tiburtina station, Italian railway officials said<br>(2) Sally Baldwin, from York, was killed instantly when a walkway collapsed and she fell into the machinery at Tiburtina station | 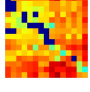 |
| N | 0.44 | (1) Bremer, 61, is a onetime assistant to former Secretaries of State William P. Rogers and Henry Kissinger and was ambassador-at-large for counterterrorism from 1986 to 1989<br>(2) Bremer, 61, is a former assistant to former Secretaries of State William P. Rogers and Henry Kissinger | 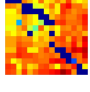 |
| N | 0.11 | (1) The initial report was made to Modesto Police December 28<br>(2) It stems from a Modesto police report | 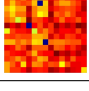 |

Table 4: Examples of sentence pairs with: ground truth labels L (P - Paraphrase, N - Not Paraphrase), the probabilities our model assigns to them ($Pr(S_1, S_2) > 0.5$ is assigned the label Paraphrase) and their similarity matrices after dynamic min-pooling. Simple paraphrase pairs have clear diagonal structure due to perfect word matches with Euclidean distance 0 (dark blue). That structure is preserved by our min-pooling layer. Best viewed in color. See text for details.

of the remaining entries. There are two shortcomings of such methods: They ignore (i) the syntactic structure of the sentences (by comparing only single words) and (ii) the global structure of such a similarity matrix (by computing only the mean).

Instead of comparing only single words [21] adds features from dependency parses. Most recently, Das and Smith [15] adopted the idea that paraphrases have related syntactic structure. Their quasi-synchronous grammar formalism incorporates a variety of features from WordNet, a named entity recognizer, a part-of-speech tagger, and the dependency labels from the aligned trees. In order to obtain high performance they combine their parsing-based model with a logistic regression model that uses 18 hand-designed surface features.

We merge these word-based models and syntactic models in one joint framework: Our matrix consists of phrase similarities and instead of just taking the mean of the similarities we can capture the global layout of the matrix via our min-pooling layer.

The idea of applying an autoencoder in a recursive setting was introduced by Pollack [9] and extended recently by [10]. Pollack's recursive auto-associative memories are similar to ours in that they are a connectionist, feedforward model. One of the major shortcomings of previous applications of recursive autoencoders to natural language sentences was their binary word representation as discussed in Sec. 2.1. Recently, Bottou discussed related ideas of recursive autoencoders [25] and recursive image and text understanding but without experimental results. Larochelle [26] investigated autoencoders with an unfolded "deep objective". Supervised recursive neural networks have been used for parsing images and natural language sentences by Socher et al. [27, 28]. Lastly, [12] introduced the standard recursive autoencoder as mentioned in Sect. 2.2.

## 6    Conclusion

We introduced an unsupervised feature learning algorithm based on unfolding, recursive autoencoders. The RAE captures syntactic and semantic information as shown qualitatively with nearest neighbor embeddings and quantitatively on a paraphrase detection task. Our RAE phrase features allow us to compare both single word vectors as well as phrases and complete syntactic trees. In order to make use of the global comparison of variable length sentences in a similarity matrix we introduce a new dynamic pooling architecture that produces a fixed-sized representation. We show that this pooled representation captures enough information about the sentence pair to determine the paraphrase relationship on the MSRP dataset with a higher accuracy than any previously published results.

## Footnotes

[1]The partitions will only be of equal size if $2n - 1$ and $2m - 1$ are divisible by $n_p$. We account for this in the following way, although many alternatives are possible. Let the number of rows of $\mathcal{S}$ be $R = 2n - 1$. Each pooling window then has $\lfloor R/n_p \rfloor$ many rows. Let $M = R \mod n_p$, be the number of remaining rows. We then evenly distribute these extra rows to the last $M$ window regions which will have $\lfloor R/n_p \rfloor + 1$ rows. The same procedure applies to the number of columns for the windows. This procedure will have a slightly finer granularity for the single word similarities which is desired for our task since word overlap is a good indicator for paraphrases. In the rare cases when $n_p > R$, the pooling layer needs to first up-sample. We achieve this by simply duplicating pixels row-wise until $R \geq n_p$.

# References

[1] E. Marsi and E. Krahmer. Explorations in sentence fusion. In *European Workshop on Natural Language Generation*, 2005.

[2] P. Clough, R. Gaizauskas, S. S. L. Piao, and Y. Wilks. METER: MEasuring TExt Reuse. In *ACL*, 2002.

[3] C. Callison-Burch. Syntactic constraints on paraphrases extracted from parallel corpora. In *Proceedings of EMNLP*, pages 196–205, 2008.

[4] B. Dolan, C. Quirk, and C. Brockett. Unsupervised construction of large paraphrase corpora: exploiting massively parallel news sources. In *COLING*, 2004.

[5] Y. Bengio, R. Ducharme, P. Vincent, and C. Janvin. A neural probabilistic language model. *J. Mach. Learn. Res.*, 3, March 2003.

[6] R. Collobert and J. Weston. A unified architecture for natural language processing: deep neural networks with multitask learning. In *ICML*, 2008.

[7] Y. Bengio, J. Louradour, Collobert R, and J. Weston. Curriculum learning. In *ICML*, 2009.

[8] J. Turian, L. Ratinov, and Y. Bengio. Word representations: a simple and general method for semi-supervised learning. In *Proceedings of ACL*, pages 384–394, 2010.

[9] J. B. Pollack. Recursive distributed representations. *Artificial Intelligence*, 46, November 1990.

[10] T. Voegtlin and P. Dominey. Linear Recursive Distributed Representations. *Neural Networks*, 18(7), 2005.

[11] J. L. Elman. Distributed representations, simple recurrent networks, and grammatical structure. *Machine Learning*, 7(2-3), 1991.

[12] R. Socher, J. Pennington, E. H. Huang, A. Y. Ng, and C. D. Manning. Semi-Supervised Recursive Autoencoders for Predicting Sentiment Distributions. In *EMNLP*, 2011.

[13] C. Goller and A. Küchler. Learning task-dependent distributed representations by backpropagation through structure. In *Proceedings of the International Conference on Neural Networks (ICNN-96)*, 1996.

[14] D. Klein and C. D. Manning. Accurate unlexicalized parsing. In *ACL*, 2003.

[15] D. Das and N. A. Smith. Paraphrase identification as probabilistic quasi-synchronous recognition. In *In Proc. of ACL-IJCNLP*, 2009.

[16] V. Rus, P. M. McCarthy, M. C. Lintean, D. S. McNamara, and A. C. Graesser. Paraphrase identification with lexico-syntactic graph subsumption. In *FLAIRS Conference*, 2008.

[17] R. Mihalcea, C. Corley, and C. Strapparava. Corpus-based and Knowledge-based Measures of Text Semantic Similarity. In *Proceedings of the 21st National Conference on Artificial Intelligence - Volume 1*, 2006.

[18] A. Islam and D. Inkpen. Semantic Similarity of Short Texts. In *Proceedings of the International Conference on Recent Advances in Natural Language Processing (RANLP 2007)*, 2007.

[19] L. Qiu, M. Kan, and T. Chua. Paraphrase recognition via dissimilarity significance classification. In *EMNLP*, 2006.

[20] S. Fernando and M. Stevenson. A semantic similarity approach to paraphrase detection. *Proceedings of the 11th Annual Research Colloquium of the UK Special Interest Group for Computational Linguistics*, 2008.

[21] S. Wan, M. Dras, R. Dale, and C. Paris. Using dependency-based features to take the "para-farce" out of paraphrase. In *Proceedings of the Australasian Language Technology Workshop 2006*, 2006.

[22] R. Barzilay and L. Lee. Learning to paraphrase: an unsupervised approach using multiple-sequence alignment. In *NAACL*, 2003.

[23] Y. Zhang and J. Patrick. Paraphrase identification by text canonicalization. In *Proceedings of the Australasian Language Technology Workshop 2005*, 2005.

[24] Z. Kozareva and A. Montoyo. Paraphrase Identification on the Basis of Supervised Machine Learning Techniques. In *Advances in Natural Language Processing, 5th International Conference on NLP, FinTAL*, 2006.

[25] L. Bottou. From machine learning to machine reasoning. *CoRR*, abs/1102.1808, 2011.

[26] H. Larochelle, Y. Bengio, J. Louradour, and P. Lamblin. Exploring strategies for training deep neural networks. *JMLR*, 10, 2009.

[27] R. Socher, C. D. Manning, and A. Y. Ng. Learning continuous phrase representations and syntactic parsing with recursive neural networks. In *Proceedings of the NIPS-2010 Deep Learning and Unsupervised Feature Learning Workshop*, 2010.

[28] R. Socher, C. Lin, A. Y. Ng, and C.D. Manning. Parsing Natural Scenes and Natural Language with Recursive Neural Networks. In *ICML*, 2011.

